# Efficient multiple hyperparameter learning for log-linear models

**Chuong B. Do**  **Chuan-Sheng Foo**  **Andrew Y. Ng**
Computer Science Department
Stanford University
Stanford, CA 94305
{chuongdo,csfoo,ang}@cs.stanford.edu

## Abstract

In problems where input features have varying amounts of noise, using distinct regularization hyperparameters for different features provides an effective means of managing model complexity. While regularizers for neural networks and support vector machines often rely on multiple hyperparameters, regularizers for structured prediction models (used in tasks such as sequence labeling or parsing) typically rely only on a single shared hyperparameter for all features. In this paper, we consider the problem of choosing regularization hyperparameters for log-linear models, a class of structured prediction probabilistic models which includes conditional random fields (CRFs). Using an implicit differentiation trick, we derive an efficient gradient-based method for learning Gaussian regularization priors with multiple hyperparameters. In both simulations and the real-world task of computational RNA secondary structure prediction, we find that multiple hyperparameter learning can provide a significant boost in accuracy compared to using only a single regularization hyperparameter.

## 1 Introduction

In many supervised learning methods, overfitting is controlled through the use of regularization penalties for limiting model complexity. The effectiveness of penalty-based regularization for a given learning task depends not only on the type of regularization penalty used (e.g., $L_1$ vs $L_2$) [29] but also (and perhaps even more importantly) on the choice of hyperparameters governing the regularization penalty (e.g., the hyperparameter $\lambda$ in an isotropic Gaussian parameter prior, $\lambda||\mathbf{w}||^2$).

When only a single hyperparameter must be tuned, cross-validation provides a simple yet reliable procedure for hyperparameter selection. For example, the regularization hyperparameter $C$ in a support vector machine (SVM) is usually tuned by training the SVM with several different values of $C$, and selecting the one that achieves the best performance on a holdout set. In many situations, using multiple hyperparameters gives the distinct advantage of allowing models with features of varying strength; for instance, in a natural language processing (NLP) task, features based on word bigrams are typically noisier than those based on individual word occurrences, and hence should be "more regularized" to prevent overfitting. Unfortunately, for sophisticated models with multiple hyperparameters [23], the naïve grid search strategy of directly trying out possible combinations of hyperparameter settings quickly grows infeasible as the number of hyperparameters becomes large.

Scalable strategies for cross-validation–based hyperparameter learning that rely on computing the gradient of cross-validation loss with respect to the desired hyperparameters arose first in the neural network modeling community [20, 21, 1, 12]. More recently, similar cross-validation optimization techniques have been proposed for other supervised learning models [3], including support vector machines [4, 10, 16], Gaussian processes [35, 33], and related kernel learning methods [18, 17, 39]. Here, we consider the problem of hyperparameter learning for a specialized class of structured classification models known as *conditional log-linear models* (CLLMs), a generalization of *conditional random fields* (CRFs) [19].

Whereas standard binary classification involves mapping an object $x \in \mathcal{X}$ to some binary output $y \in \mathcal{Y}$ (where $\mathcal{Y} = \{\pm 1\}$), the input space $\mathcal{X}$ and output space $\mathcal{Y}$ in a structured classification task generally contain complex combinatorial objects (such as sequences, trees, or matchings). Designing hyperparameter learning algorithms for structured classification models thus yields a number of unique computational challenges not normally encountered in the flat classification setting. In this paper, we derive a gradient-based approach for optimizing the hyperparameters of a CLLM using the loss incurred on a holdout set. We describe the required algorithms specific to CLLMs which make the needed computations tractable. Finally, we demonstrate on both simulations and a real-world computational biology task that our hyperparameter learning method can give gains over learning flat unstructured regularization priors.

## 2 Preliminaries

Conditional log-linear models (CLLMs) are a probabilistic framework for sequence labeling or parsing problems, where $\mathcal{X}$ is an exponentially large space of possible input sequences and $\mathcal{Y}$ is an exponentially large space of candidate label sequences or parse trees. Let $\mathbf{F} : \mathcal{X} \times \mathcal{Y} \to \mathbb{R}^n$ be a fixed vector-valued mapping from input-output pairs to an $n$-dimensional feature space. CLLMs model the conditional probability of $y$ given $x$ as $P(y \mid x; \mathbf{w}) = \exp(\mathbf{w}^T \mathbf{F}(x, y))/Z(\mathbf{x})$ where $Z(\mathbf{x}) = \sum_{y' \in \mathcal{Y}} \exp(\mathbf{w}^T \mathbf{F}(x, y'))$. Given a training set $T = \left\{ (x^{(i)}, y^{(i)}) \right\}_{i=1}^m$ of i.i.d. labeled input-output pairs drawn from some unknown fixed distribution $\mathcal{D}$ over $\mathcal{X} \times \mathcal{Y}$, the parameter learning problem is typically posed as *maximum a posteriori* (MAP) estimation (or equivalently, regularized logloss minimization):

$$\mathbf{w}^\star = \underset{\mathbf{w} \in \mathbb{R}^n}{\arg\min} \left( \frac{1}{2} \mathbf{w}^T \mathbf{C} \mathbf{w} - \sum_{i=1}^m \log P(y^{(i)} \mid x^{(i)}; \mathbf{w}) \right), \qquad \text{(OPT1)}$$

where $\frac{1}{2}\mathbf{w}^T \mathbf{C} \mathbf{w}$ (for some positive definite matrix $\mathbf{C}$) is a regularization penalty used to prevent overfitting. Here, $\mathbf{C}$ is the inverse covariance matrix of a Gaussian prior on the parameters $\mathbf{w}$.

While a number of efficient procedures exist for solving the optimization problem OPT1 [34, 11], little attention is usually given to choosing an appropriate regularization matrix $\mathbf{C}$. Generally, $\mathbf{C}$ is parameterized using a small number of free variables, $\mathbf{d} \in \mathbb{R}^k$, known as the *hyperparameters* of the model. Given a holdout set $H = \left\{ (\tilde{x}^{(i)}, \tilde{y}^{(i)}) \right\}_{i=1}^{\tilde{m}}$ of i.i.d. examples drawn from $\mathcal{D}$, hyperparameter learning itself can be cast as an optimization problem:

$$\underset{\mathbf{d} \in \mathbb{R}^k}{\text{minimize}} \quad -\sum_{i=1}^{\tilde{m}} \log P\Big( \tilde{y}^{(i)} \mid \tilde{x}^{(i)}; \mathbf{w}^\star(\mathbf{C}) \Big). \qquad \text{(OPT2)}$$

In words, OPT2 finds the hyperparameters $\mathbf{d}$ whose regularization matrix $\mathbf{C}$ leads the parameter vector $\mathbf{w}^\star(\mathbf{C})$ learned from the training set to obtain small logloss on holdout data. For many real-world applications, $\mathbf{C}$ is assumed to take a simple form, such as a scaled identity matrix, $C\mathbf{I}$. While this parameterization may be partially motivated by concerns of hyperparameter overfitting [28], such a choice usually stems from the difficulty of hyperparameter inference.

In practice, grid-search procedures provide a reliable method for determining hyperparameters to low-precision: one trains the model using several candidate values of $C$ (e.g., $C \in \left\{ \dots, 2^{-2}, 2^{-1}, 2^0, 2^1, 2^2, \dots \right\}$), and chooses the $C$ that minimizes holdout logloss. While this strategy is suitable for tuning a single model hyperparameter, more sophisticated strategies are necessary when optimizing multiple hyperparameters.

## 3 Learning multiple hyperparameters

In this section, we lay the framework for multiple hyperparameter learning by describing a simple yet flexible parameterization of $\mathbf{C}$ that arises quite naturally in many practical problems. We then describe a generic strategy for hyperparameter adaptation via gradient-based optimization.

Consider a setting in which predefined subsets of parameter components (which we call *regularization groups*) are constrained to use the same hyperparameters [6]. For instance, in an NLP task, individual word occurrence features may be placed in a separate regularization group from word bigram features. Formally, let $k$ be a fixed number of regularization groups, and let $\pi : \{1, \dots, n\} \to \{1, \dots, k\}$ be a prespecified mapping from parameters to regularization groups. Furthermore, for a vector $\mathbf{x} \in \mathbb{R}^k$, define its expansion $\overline{\mathbf{x}} \in \mathbb{R}^n$ as $\overline{\mathbf{x}} = (x_{\pi(1)}, x_{\pi(2)}, \dots, x_{\pi(n)})$.

In the sequel, we parameterize $\mathbf{C} \in \mathbb{R}^{n \times n}$ in terms of some hyperparameter vector $\mathbf{d} \in \mathbb{R}^k$ as the diagonal matrix, $\mathbf{C}(\mathbf{d}) = \mathbf{diag}(\exp(\overline{\mathbf{d}}))$. Under this representation, $\mathbf{C}(\mathbf{d})$ is necessar-

ily positive definite, so OPT2 can be written as an unconstrained minimization over the variables $\mathbf{d} \in \mathbb{R}^k$. Specifically, let $\ell_T(\mathbf{w}) = -\sum_{i=1}^m \log P\big(y^{(i)} \mid x^{(i)}; \mathbf{w}\big)$ denote the training logloss and $\ell_H(\mathbf{w}) = -\sum_{i=1}^{\tilde{m}} \log P\big(\tilde{y}^{(i)} \mid \tilde{x}^{(i)}; \mathbf{w}\big)$ the holdout logloss for a parameter vector $\mathbf{w}$. Omitting the dependence of $\mathbf{C}$ on $\mathbf{d}$ for notational convenience, we have the optimization problem

$$\underset{\mathbf{d} \in \mathbb{R}^k}{\text{minimize}} \quad \ell_H(\mathbf{w}^\star) \qquad \text{subject to} \quad \mathbf{w}^\star = \underset{\mathbf{w} \in \mathbb{R}^n}{\arg\min} \left( \frac{1}{2} \mathbf{w}^T \mathbf{C} \mathbf{w} + \ell_T(\mathbf{w}) \right). \qquad \text{(OPT2')}$$

For any fixed setting of these hyperparameters, the objective function of OPT2' can be evaluated by (1) using the hyperparameters $\mathbf{d}$ to determine the regularization matrix $\mathbf{C}$, (2) solving OPT1 using $\mathbf{C}$ to determine $\mathbf{w}^\star$ and (3) computing the holdout logloss using the parameters $\mathbf{w}^\star$. In this next section, we derive a method for computing the gradient of the objective function of OPT2' with respect to the hyperparameters. Given both procedures for function and gradient evaluation, we may apply standard gradient-based optimization (e.g., conjugate gradient or L-BFGS [30]) in order to find a local optimum of the objective. In general, we observe that only a few iterations ($\sim 5$) are usually sufficient to determine reasonable hyperparameters to low accuracy.

## 4 The hyperparameter gradient

Note that the optimization objective $\ell_H(\mathbf{w}^\star)$ is a function of $\mathbf{w}^\star$. In turn, $\mathbf{w}^\star$ is a function of the hyperparameters $\mathbf{d}$, as implicitly defined by the gradient stationarity condition, $\mathbf{C}\mathbf{w}^\star + \nabla_\mathbf{w} \ell_T(\mathbf{w}^\star) = \mathbf{0}$. To compute the hyperparameter gradient, we will use both of these facts.

### 4.1 Deriving the hyperparameter gradient

First, we apply the chain rule to the objective function of OPT2' to obtain

$$\nabla_\mathbf{d} \ell_H(\mathbf{w}^\star) = \mathbf{J}_\mathbf{d}^T \nabla_\mathbf{w} \ell_H(\mathbf{w}^\star) \qquad (1)$$

where $\mathbf{J}_\mathbf{d}$ is the $n \times k$ Jacobian matrix whose $(i,j)$th entry is $\partial w_i^\star / \partial d_j$. The term $\nabla_\mathbf{w} \ell_H(\mathbf{w}^\star)$ is simply the gradient of the holdout logloss evaluated at $\mathbf{w}^\star$. For decomposable models, this may be computed exactly via dynamic programming (e.g., the forward/backward algorithm for chain-structured models or the inside/outside algorithm for grammar-based models).

Next, we show how to compute the Jacobian matrix $\mathbf{J}_\mathbf{d}$. Recall that at the optimum of the smooth unconstrained optimization problem OPT1, the partial derivative of the objective with respect to any parameter must vanish. In particular, the partial derivative of $\frac{1}{2}\mathbf{w}^T \mathbf{C} \mathbf{w} + \ell_T(\mathbf{w})$ with respect to $w_i$ vanishes when $\mathbf{w} = \mathbf{w}^\star$, so

$$0 = \mathbf{C}_i^T \mathbf{w}^\star + \frac{\partial}{\partial w_i} \ell_T(\mathbf{w}^\star), \qquad (2)$$

where $\mathbf{C}_i^T$ denotes the $i$th row of the $\mathbf{C}$ matrix. Since (2) uniquely defines $\mathbf{w}^\star$ (as OPT1 is a strictly convex optimization problem), we can use implicit differentiation to obtain the needed partial derivatives. Specifically, we can differentiate both sides of (2) with respect to $d_j$ to obtain

$$0 = \sum_{p=1}^n \left( w_p^\star \frac{\partial}{\partial d_j} C_{ip} + C_{ip} \frac{\partial}{\partial d_j} w_p^\star \right) + \sum_{p=1}^n \frac{\partial}{\partial w_p} \frac{\partial}{\partial w_i} \ell_T(\mathbf{w}^\star) \frac{\partial}{\partial d_j} w_p^\star, \qquad (3)$$

$$= \mathbf{I}_{\{\pi(i)=j\}} w_i^\star \exp(d_j) + \sum_{p=1}^n \left( C_{ip} + \frac{\partial}{\partial w_p} \frac{\partial}{\partial w_i} \ell_T(\mathbf{w}^\star) \right) \frac{\partial}{\partial d_j} w_p^\star. \qquad (4)$$

Stacking (4) for all $i \in \{1, \ldots, n\}$ and $j \in \{1, \ldots, k\}$, we obtain the equivalent matrix equation,

$$\mathbf{0} = \mathbf{B} + (\mathbf{C} + \nabla_\mathbf{w}^2 \ell_T(\mathbf{w}^\star)) \mathbf{J}_\mathbf{d} \qquad (5)$$

where $\mathbf{B}$ is the $n \times k$ matrix whose $(i,j)$th element is $\mathbf{I}_{\{\pi(i)=j\}} w_i^\star \exp(d_j)$, and $\nabla_\mathbf{w}^2 \ell_T(\mathbf{w}^\star)$ is the Hessian of the training logloss evaluated at $\mathbf{w}^\star$. Finally, solving these equations for $\mathbf{J}_\mathbf{d}$, we obtain

$$\mathbf{J}_\mathbf{d} = -(\mathbf{C} + \nabla_\mathbf{w}^2 \ell_T(\mathbf{w}^\star))^{-1} \mathbf{B}. \qquad (6)$$

### 4.2 Computing the hyperparameter gradient efficiently

In principle, one could simply use (6) to obtain the Jacobian matrix $\mathbf{J}_\mathbf{d}$ directly. However, computing the $n \times n$ matrix $(\mathbf{C} + \nabla_\mathbf{w}^2 \ell_T(\mathbf{w}^\star))^{-1}$ is difficult. Computing the Hessian matrix $\nabla_\mathbf{w}^2 \ell_T(\mathbf{w}^\star)$ in a typical CLLM requires approximately $n$ times the cost of a single logloss gradient evaluation. Once the Hessian has been computed, typical matrix inversion routines take $O(n^3)$ time. Even more problematic, the $\Omega(n^2)$ memory usage for storing the Hessian is prohibitive as typical log-linear models (e.g., in NLP) may have thousands or even millions of features. To deal with these

---

**Algorithm 1**: Gradient computation for hyperparameter selection.

---

Input:  training set $T = \left\{(x^{(i)}, y^{(i)})\right\}_{i=1}^{m}$, holdout set $H = \left\{(\tilde{x}^{(i)}, \tilde{y}^{(i)})\right\}_{i=1}^{\tilde{m}}$
current hyperparameters $\mathbf{d} \in \mathbb{R}^{k}$

Output:  hyperparameter gradient $\nabla_{\mathbf{d}} \ell_H(\mathbf{w}^{\star})$

1. Compute solution $\mathbf{w}^{\star}$ to OPT1 using regularization matrix $\mathbf{C} = \mathbf{diag}(\mathbf{exp}(\overline{\mathbf{d}}))$.
2. Form the matrix $\mathbf{B} \in \mathbb{R}^{n \times k}$ such that $(\mathbf{B})_{ij} = \mathbf{I}_{\{\pi(i)=j\}} w_i^{\star} \exp(d_j)$.
3. Use conjugate gradient algorithm to solve the linear system,
$$(\mathbf{C} + \nabla_{\mathbf{w}}^2 \ell_T(\mathbf{w}^{\star}))\mathbf{x} = \nabla_{\mathbf{w}} \ell_H(\mathbf{w}^{\star}).$$
4. Return $-\mathbf{B}^T \mathbf{x}$.

---

Figure 1: Pseudocode for gradient computation

problems, we first explain why $(\mathbf{C} + \nabla_{\mathbf{w}}^2 \ell_T(\mathbf{w}^{\star}))\mathbf{v}$ for any arbitrary vector $\mathbf{v} \in \mathbb{R}^n$ can be computed in $O(n)$ time, even though forming $(\mathbf{C} + \nabla_b^2 w \ell_T(\mathbf{w}^{\star}))^{-1}$ is expensive. Using this result, we then describe an efficient procedure for computing the holdout hyperparameter gradient which avoids the expensive Hessian computation and inversion steps of the direct method.

First, since $\mathbf{C}$ is diagonal, the product of $\mathbf{C}$ with any arbitrary vector $\mathbf{v}$ is trivially computable in $O(n)$ time. Second, although direct computation of the Hessian is inefficient in a generic log-linear model, computing the product of the Hessian with $\mathbf{v}$ can be done quickly, using any of the following techniques, listed in order of increasing implementation effort (and numerical precision):

1. **Finite differencing**. Use the following numerical approximation:
$$\nabla_{\mathbf{w}}^2 \ell_T(\mathbf{w}^{\star}) \cdot \mathbf{v} = \lim_{r \to 0} \frac{\nabla_{\mathbf{w}} \ell_T(\mathbf{w}^{\star} + r\mathbf{v}) - \nabla_{\mathbf{w}} \ell_t(\mathbf{w}^{\star})}{r}. \tag{7}$$

2. **Complex step derivative** [24]. Use the following identity from complex analysis:
$$\nabla_{\mathbf{w}}^2 \ell_T(\mathbf{w}^{\star}) \cdot \mathbf{v} = \lim_{r \to 0} \frac{\mathrm{Im}\left\{\nabla_{\mathbf{w}} \ell_T(\mathbf{w}^{\star} + i \cdot r\mathbf{v})\right\}}{r}. \tag{8}$$

   where $\mathrm{Im}\{\cdot\}$ denotes the imaginary part of its complex argument (in this case, a vector). Because there is no subtraction in the numerator of the right-hand expression, the complex-step derivative does not suffer from the numerical problems of the finite-differencing method that result from cancellation. As a consequence, much smaller step sizes can be used, allowing for greater accuracy.

3. **Analytical computation**. Given an existing $O(n)$ algorithm for computing gradients analytically, define the differential operator
$$\mathcal{R}_{\mathbf{v}}\{f(\mathbf{w})\} = \lim_{r \to 0} \frac{f(\mathbf{w} + r\mathbf{v}) - f(\mathbf{w})}{r} = \left.\frac{\partial}{\partial r} f(\mathbf{w} + r\mathbf{v})\right|_{r=0}, \tag{9}$$

   for which one can verify that $\mathcal{R}_{\mathbf{v}}\{\nabla_{\mathbf{w}} \ell_T(\mathbf{w}^{\star})\} = \nabla_{\mathbf{w}}^2 \ell_T(\mathbf{w}^{\star}) \cdot \mathbf{v}$. By applying standard rules for differential operators, $\mathcal{R}_{\mathbf{v}}\{\nabla_{\mathbf{w}} \ell_T(\mathbf{w}^{\star})\}$ can be computed recursively using a modified version of the original gradient computation routine; see [31] for details.

Hessian-vector products for graphical models were previously used in the context of step-size adaptation for stochastic gradient descent [36]. In our experiments, we found that the simplest method, finite-differencing, provided sufficient accuracy for our application.

Given the above procedure for computing matrix-vector products, we can now use the *conjugate gradient* (CG) method to solve the matrix equation (5) to obtain $\mathbf{J_d}$. Unlike direct methods for solving linear systems $\mathbf{Ax} = \mathbf{b}$, CG is an iterative method which relies on the matrix $\mathbf{A}$ only through matrix-vector products $\mathbf{Av}$. In practice, few steps of the CG algorithm are generally needed to find an approximate solution of a linear system with acceptable accuracy. Using CG in this way amounts to solving $k$ linear systems, one for each column of the $\mathbf{J_d}$ matrix. Unlike the direct method of forming the $(\mathbf{C} + \nabla_{\mathbf{w}}^2 \ell_T(\mathbf{w}^{\star}))$ matrix and its inverse, solving the linear systems avoids the expensive $\Omega(n^2)$ cost of Hessian computation and matrix inversion.

Nevertheless, even this approach for computing the Jacobian matrices still requires the solution of multiple linear systems, which scales poorly when the number of hyperparameters $k$ is large.

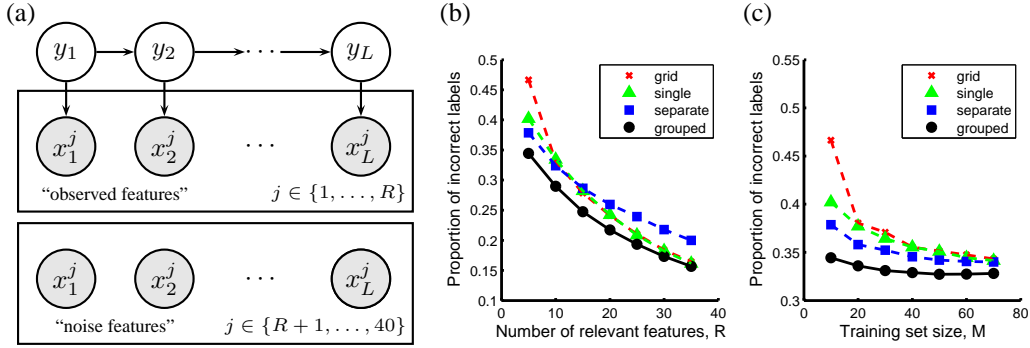

Figure 2: HMM simulation experiments. (a) State diagram of the HMM used in the simulations. (b) Testing set performance when varying $R$, using $M = 10$. (c) Testing set performance when varying $M$, using $R = 5$. In both (b) and (c), each point represents an average over 100 independent runs of HMM training/holdout/testing set generation and CRF training and hyperparameter optimization.

However, we can do much better by reorganizing the computations in such a way that the Jacobian matrix $\mathbf{J}_d$ is never explicitly required. In particular, substituting (6) into (1),

$$\nabla_{\mathbf{d}}\ell_H(\mathbf{w}^\star) = -\mathbf{B}^T(\mathbf{C} + \nabla_{\mathbf{w}}^2\ell_T(\mathbf{w}^\star))^{-1}\nabla_{\mathbf{w}}\ell_H(\mathbf{w}^\star) \tag{10}$$

we observe that it suffices to solve the single linear system,

$$(\mathbf{C} + \nabla_{\mathbf{w}}^2\ell_T(\mathbf{w}^\star))\mathbf{x} = \nabla_{\mathbf{w}}\ell_H(\mathbf{w}^\star) \tag{11}$$

and then form $\nabla_{\mathbf{d}}\ell_H(\mathbf{w}^\star) = -\mathbf{B}^T\mathbf{x}$. By organizing the computations this way, the number of least squares problems that must be solved is substantially reduced from $k$ to only one. A similar trick was previously used for hyperparameter adaptation in SVMs [16] and kernel logistic regression [33]. Figure 1 shows a summary of our algorithm for hyperparameter gradient computation.[1]

## 5  Experiments

To test the effectiveness of our hyperparameter learning algorithm, we applied it to two tasks: a simulated sequence labeling task involving noisy features, and a real-world application of conditional log-linear models to the biological problem of RNA secondary structure prediction.

**Sequence labeling simulation.** For our simulation test, we constructed a simple linear-chain hidden Markov model (HMM) with binary-valued hidden nodes, $y_i \in \{0, 1\}$.[2] We associated 40 binary-valued features $x_i^j$, $j \in \{1, \ldots, 40\}$ with each hidden state $y_i$, including $R$ "relevant" observed features whose values were chosen based on $y_i$, and $(40 - R)$ "irrelevant" noise features whose values were chosen to be either 0 or 1 with equal probability, independent of $y_i$.[3] Figure 2a shows the graphical model representing the HMM. For each run, we used the HMM to simulate training, holdout, and testing sets of $M$, 10, and 1000 sequences, respectively, each of length 10.

Next, we constructed a CRF based on an HMM model similar to that shown in Figure 2a in which potentials were included for the initial node $y_1$, between each $y_i$ and $y_{i+1}$, and between $y_i$ and each $x_i^j$ (including both the observed features and the noise features). We then performed gradient-based hyperparameter learning using three different parameter-tying schemes: (a) all hyperparameters constrained to be equal, (b) separate hyperparameter groups for each parameter of the model, and (c) transitions, observed features, and noise features each grouped together. Figure 2b shows the performance of the CRF for each of the three parameter-tying gradient-based optimization schemes, as well as the performance of scheme (a) when using the standard grid-search strategy of trying regularization matrices $C\mathbf{I}$ for $C \in \{\ldots, 2^{-2}, 2^{-1}, 2^0, 2^1, 2^2, \ldots\}$.

As seen in Figures 2b and 2c, the gradient-based procedure performed either as well as or better than a grid search for single hyperparameter models. Using either a single hyperparameter or all separate hyperparameters generally gave similar results, with a slight tendency for the separate

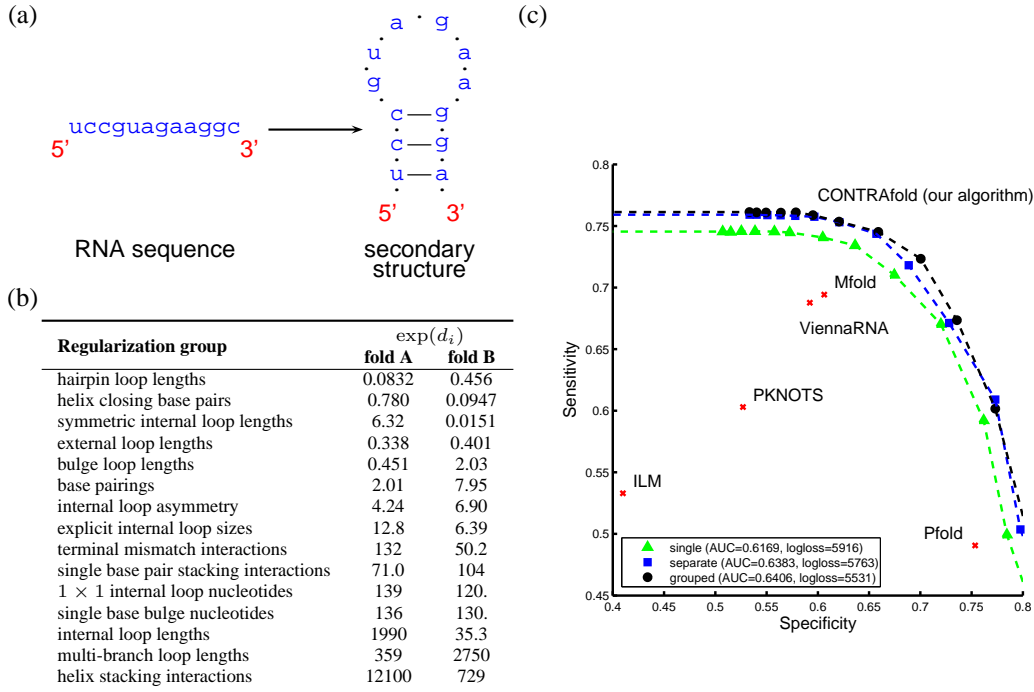

Figure 3: RNA secondary structure prediction. (a) An illustration of the secondary structure prediction task. (b) Grouped hyperparameters learned using our algorithm for each of the two folds. (c) Performance comparison with state-of-the-art methods when using either a single hyperparameter (the "original" CONTRAfold), separate hyperparameters, or grouped hyperparameters.

hyperparameter model to overfit. Enforcing regularization groups, however, gave consistently lower error rates, achieving an absolute reduction in generalization error over the next-best model of 6.7%, corresponding to a relative reduction of 16.2%.

**RNA secondary structure prediction.** We also applied our framework to the problem of RNA secondary structure prediction. Ribonucleic acid (RNA) molecules are long nucleic acid polymers present in the cells of all living organisms. For many types of RNA, three-dimensional (or tertiary) structure plays an important role in determining the RNA's function. Here, we focus on the task of predicting RNA secondary structure, i.e., the pattern of nucleotide base pairings which form the two-dimensional scaffold upon which RNA tertiary structures assemble (see Figure 3a).

As a starting point, we used CONTRAfold [7], a current state-of-the-art secondary structure prediction program based on CLLMs. In brief, the CONTRAfold program models RNA secondary structures using a variant of stochastic context-free grammars (SCFGs) which incorporates features chosen to closely match the energetic terms found in standard physics-based models of RNA structure. These features model the various types of loops that occur in RNAs (e.g., hairpin loops, bulge loops, interior loops, etc.). To control overfitting, CONTRAfold uses flat $L_2$ regularization. Here, we modified the existing implementation to perform an "outer" optimization loop based on our algorithm, and chose regularization groups either by (a) enforcing a single hyperparameter group, (b) using separate groups for each parameter, or (c) grouping according to the type of each feature (e.g., all features for describing hairpin loop lengths were placed in a single regularization group).

For testing, we collected 151 RNA sequences from the Rfam database [13] for which experimentally-determined secondary structures were already known. We divided this dataset into two folds (denoted A and B) and performed two-fold cross-validation. Despite the small size of the training set, the hyperparameters learned on each fold were nonetheless qualitatively similar, indicating the robustness of the procedure (see Figure 3b). As expected, features with small regularization hyperparameters correspond to properties of RNAs which are known to contribute strongly to the energetics of RNA secondary structure, whereas many of the features with larger regularization hyperparameters indicate structural properties whose presence/absence are either less correlated with RNA secondary structure or sufficiently noisy that their parameters are difficult to determine reliably from the training data.

We then compared the cross-validated performance of algorithm with state-of-the-art methods (see Figure 3c).[4] Using separate or grouped hyperparameters both gave increased sensitivity and increased specificity compared to the original model, which was learned using a single regularization hyperparameter. Overall, the testing logloss (summed over the two folds) decreased by roughly 6.5% when using grouped hyperparameters and 2.6% when using multiple separate hyperparameters, while the estimated testing ROC area increased by roughly 3.8% and 3.4%, respectively.

## 6 Discussion and related work

In this work, we presented a gradient-based approach for hyperparameter learning based on minimizing logloss on a holdout set. While the use of cross-validation loss as a proxy for generalization error is fairly natural, in many other supervised learning methods besides log-linear models, other objective functions have been proposed for hyperparameter optimization. In SVMs, approaches based on optimizing generalization bounds [4], such as the radius/margin-bound [15] or maximal discrepancy criterion [2] have been proposed. Comparable generalization bounds are not generally known for CRFs; even in SVMs, however, generalization bound-based methods empirically do not outperform simpler methods based on optimizing five-fold cross-validation error [8].

A different method for dealing with hyperparameters, common in neural network modeling, is the Bayesian approach of treating hyperparameters themselves as parameters in the model to be estimated. In an ideal Bayesian scheme, one does not perform hyperparameter or parameter inference, but rather integrates over all possible hyperparameters and parameters in order to obtain a posterior distribution over predicted outputs given the training data. This integration can be performed using a hybrid Monte Carlo strategy [27, 38]. For the types of large-scale log-linear models we consider in this paper, however, the computational expense of sampling-based strategies can be extremely high due to slow convergence of MCMC techniques [26].

Empirical Bayesian (i.e., ML-II) strategies, such as Automatic Relevance Determination (ARD) [22], take the intermediate approach of integrating over parameters to obtain the marginal likelihood (known as the log evidence), which is then optimized with respect to the hyperparameters. Computing marginal likelihoods, however, can be quite costly, especially for log-linear models. One method for doing this involves approximating the parameter posterior distribution as a Gaussian centered at the posterior mode [22, 37]. In this strategy, however, the "Occam factor" used for hyperparameter optimization still requires a Hessian computation, which does not scale well for log-linear models. An alternate approach based on using a modification of expectation propagation (EP) [25] was applied in the context of Bayesian CRFs [32] and later extended to graph-based semi-supervised learning [14]. As described, however, inference in these models relies on non-traditional "probit-style" potentials for efficiency reasons, and known algorithms for inference in Bayesian CRFs are limited to graphical models with fixed structure.

In contrast, our approach works broadly for a variety of log-linear models, including the grammar-based models common in computational biology and natural language processing. Furthermore, our algorithm is simple and efficient, both conceptually and in practice: one iteratively optimizes the parameters of a log-linear model using a fixed setting of the hyperparameters, and then one changes the hyperparameters based on the holdout logloss gradient. The gradient computation relies primarily on a simple conjugate gradient solver for linear systems, coupled with the ability to compute Hessian-vector products (straightforward in any modern programming language that allows for operation overloading). As we demonstrated in the context of RNA secondary structure prediction, gradient-based hyperparameter learning is a practical and effective method for tuning hyperparameters when applied to large-scale log-linear models.

Finally we note that for neural networks, [9] and [5] proposed techniques for simultaneous optimization of hyperparameters and parameters; these results suggest that similar procedures for faster hyperparameter learning that do not require a doubly-nested optimization may be possible.

## Footnotes

[1] In practice, roughly 50-100 iterations of CG were sufficient to obtain hyperparameter gradients, meaning that the cost of running Algorithm 1 was approximately the same as the cost of solving OPT1 for a single fixed setting of the hyperparameters. Roughly 3-5 line searches were sufficient to identify good hyperparameter settings; assuming that each line search takes 2-4 times the cost of solving OPT1, the overall hyperparameter learning procedure takes approximately 20 times the cost of solving OPT1 once.

[2] For our HMM, we set initial state probabilities to 0.5 each, and used self-transition probabilities of 0.6.

[3] Specifically, we drew each $x_i^j$ independently according to $P(x_i^j = v \mid y_i = v) = 0.6$, $v \in \{0, 1\}$.

## References

[1] L. Andersen, J. Larsen, L. Hansen, and M. Hintz-Madsen. Adaptive regularization of neural classifiers. In *NNSP*, 1997.

[2] D. Anguita, S. Ridella, F. Rivieccio, and R. Zunino. Hyperparameter design criteria for support vector classifiers. *Neurocomputing*, 55:109–134, 2003.

---

[4]Following [7], we used the maximum expected accuracy algorithm for decoding, which returns a set of candidates parses reflecting different trade-offs between sensitivity (proportion of true base-pairs called) and specificity (proportion of called base-pairs which are correct).

[3] Y. Bengio. Gradient-based optimization of hyperparameters. *Neural Computation*, 12:1889–1900, 2000.

[4] O. Chapelle, V. Vapnik, O. Bousquet, and S. Mukherjee. Choosing multiple parameters for support vector machines. *Machine Learning*, 46(1–3):131–159, 2002.

[5] D. Chen and M. Hagan. Optimal use of regularization and cross-validation in neural network modeling. In *IJCNN*, 1999.

[6] C. B. Do, S. S. Gross, and S. Batzoglou. CONTRAlign: discriminative training for protein sequence alignment. In *RECOMB*, pages 160–174, 2006.

[7] C. B. Do, D. A. Woods, and S. Batzoglou. CONTRAfold: RNA secondary structure prediction without physics-based models. *Bioinformatics*, 22(14):e90–e98, 2006.

[8] K. Duan, S. S. Keerthi, and A.N. Poo. Evaluation of simple performance measures for tuning SVM hyperparameters. *Neurocomputing*, 51(4):41–59, 2003.

[9] R. Eigenmann and J. A. Nossek. Gradient based adaptive regularization. In *NNSP*, pages 87–94, 1999.

[10] T. Glasmachers and C. Igel. Gradient-based adaptation of general Gaussian kernels. *Neural Comp.*, 17(10):2099–2105, 2005.

[11] A. Globerson, T. Y. Koo, X. Carreras, and M. Collins. Exponentiated gradient algorithms for log-linear structured prediction. In *ICML*, pages 305–312, 2007.

[12] C. Goutte and J. Larsen. Adaptive regularization of neural networks using conjugate gradient. In *ICASSP*, 1998.

[13] S. Griffiths-Jones, S. Moxon, M. Marshall, A. Khanna, S. R. Eddy, and A. Bateman. Rfam: annotating non-coding RNAs in complete genomes. *Nucleic Acids Res*, 33:D121–D124, 2005.

[14] A. Kapoor, Y. Qi, H. Ahn, and R. W. Picard. Hyperparameter and kernel learning for graph based semi-supervised classification. In *NIPS*, pages 627–634, 2006.

[15] S. S. Keerthi. Efficient tuning of SVM hyperparameters using radius/margin bound and iterative algorithms. *IEEE Transaction on Neural Networks*, 13(5):1225–1229, 2002.

[16] S. S. Keerthi, V. Sindhwani, and O. Chapelle. An efficient method for gradient-based adaptation of hyperparameters in SVM models. In *NIPS*, 2007.

[17] K. Kobayashi, D. Kitakoshi, and R. Nakano. Yet faster method to optimize SVR hyperparameters based on minimizing cross-validation error. In *IJCNN*, volume 2, pages 871–876, 2005.

[18] K. Kobayashi and R. Nakano. Faster optimization of SVR hyperparameters based on minimizing cross-validation error. In *IEEE Conference on Cybernetics and Intelligent Systems*, 2004.

[19] J. Lafferty, A. McCallum, and F. Pereira. Conditional random fields: probabilistic models for segmenting and labeling sequence data. In *ICML 18*, pages 282–289, 2001.

[20] J. Larsen, L. K. Hansen, C. Svarer, and M. Ohlsson. Design and regularization of neural networks: the optimal use of a validation set. In *NNSP*, 1996.

[21] J. Larsen, C. Svarer, L. N. Andersen, and L. K. Hansen. Adaptive regularization in neural network modeling. In *Neural Networks: Tricks of the Trade*, pages 113–132, 1996.

[22] D. J. C. MacKay. Bayesian interpolation. *Neural Computation*, 4(3):415–447, 1992.

[23] D. J. C. MacKay and R. Takeuchi. Interpolation models with multiple hyperparameters. *Statistics and Computing*, 8:15–23, 1998.

[24] J. R. R. A. Martins, P. Sturdza, and J. J. Alonso. The complex-step derivative approximation. *ACM Trans. Math. Softw.*, 29(3):245–262, 2003.

[25] T. P. Minka. Expectation propagation for approximate Bayesian inference. In *UAI*, volume 17, pages 362–369, 2001.

[26] I. Murray and Z. Ghahramani. Bayesian learning in undirected graphical models: approximate MCMC algorithms. In *UAI*, pages 392–399, 2004.

[27] R. M. Neal. *Bayesian Learning for Neural Networks*. Springer, 1996.

[28] A. Y. Ng. Preventing overfitting of cross-validation data. In *ICML*, pages 245–253, 1997.

[29] A. Y. Ng. Feature selection, $L_1$ vs. $L_2$ regularization, and rotational invariance. In *ICML*, 2004.

[30] J. Nocedal and S. J. Wright. *Numerical Optimization*. Springer, 1999.

[31] B. A. Pearlmutter. Fast exact multiplication by the Hessian. *Neural Comp*, 6(1):147–160, 1994.

[32] Y. Qi, M. Szummer, and T. P. Minka. Bayesian conditional random fields. In *AISTATS*, 2005.

[33] M. Seeger. Cross-validation optimization for large scale hierarchical classification kernel methods. In *NIPS*, 2007.

[34] F. Sha and F. Pereira. Shallow parsing with conditional random fields. In *NAACL*, pages 134–141, 2003.

[35] S. Sundararajan and S. S. Keerthi. Predictive approaches for choosing hyperparameters in Gaussian processes. *Neural Comp.*, 13(5):1103–1118, 2001.

[36] S. V. N. Vishwanathan, N. N. Schraudolph, M. W. Schmidt, and K. P. Murphy. Accelerated training of conditional random fields with stochastic gradient methods. In *ICML*, pages 969–976, 2006.

[37] M. Wellings and S. Parise. Bayesian random fields: the Bethe-Laplace approximation. In *ICML*, 2006.

[38] C. K. I. Williams and D. Barber. Bayesian classification with Gaussian processes. *IEEE Transactions on Pattern Analysis and Machine Intelligence*, 20(12):1342–1351, 1998.

[39] X. Zhang and W. S. Lee. Hyperparameter learning for graph based semi-supervised learning algorithms. In *NIPS*, 2007.
